# A Model of Distributed Sensorimotor Control in the Cockroach Escape Turn

R.D. Beer[1,2], G.J. Kacmarcik[1], R.E. Ritzmann[2] and H.J. Chiel[2]
Departments of [1]Computer Engineering and Science, and [2]Biology
Case Western Reserve University
Cleveland, OH 44106

## Abstract

In response to a puff of wind, the American cockroach turns away and runs. The circuit underlying the initial turn of this escape response consists of three populations of individually identifiable nerve cells and appears to employ distributed representations in its operation. We have reconstructed several neuronal and behavioral properties of this system using simplified neural network models and the backpropagation learning algorithm constrained by known structural characteristics of the circuitry. In order to test and refine the model, we have also compared the model's responses to various lesions with the insect's responses to similar lesions.

## 1  INTRODUCTION

It is becoming generally accepted that many behavioral and cognitive capabilities of the human brain must be understood as resulting from the cooperative activity of populations of nerve cells rather than the individual activity of any particular cell. For example, distributed representation of orientation by populations of directionally-tuned neurons appears to be a common principle of many mammalian motor control systems (Georgopoulos *et al.*, 1988; Lee *et al.*, 1988). While the general principles of distributed processing are evident in these mammalian systems, however, the details of their operation are not. Without a deeper knowledge of the underlying neuronal circuitry and its inputs and outputs, it is difficult to answer such questions as how the population code is formed, how it is read out, and what precise role it plays in the operation of the nervous system as a whole. In this paper, we describe our work with an invertebrate system, the cockroach escape response, which offers the possibility of addressing these questions.

## 2   THE COCKROACH ESCAPE RESPONSE

Any sudden puff of wind directed toward the American cockroach (*Periplaneta americana*), such as from an attacking predator, evokes a rapid directional turn away from the wind source followed by a run (Ritzmann, 1984). The initial turn is generally completed in approximately 60 msec after the onset of the wind. During this time, the insect must integrate information from hundreds of sensors to direct a very specific set of leg movements involving dozens of muscles distributed among three distinct pairs of multisegmented legs. In addition, the response is known to exhibit various forms of plasticity, including adaptation to sensory lesions. This system has also recently been shown to be capable of multiphasic responses (e.g. an attack from the front may elicit a sequence of escape movements rather than a single turn) and context-dependent responses (e.g. if the cockroach is in antennal contact with an obstacle, it may modify its escape movements accordingly) (Ritzmann *et al.*, in preparation).

The basic architecture of the neuronal circuitry responsible for the initial turn of the escape response is known (Daley and Camhi, 1988; Ritzmann and Pollack, 1988; Ritzmann and Pollack, 1990). Characteristics of the initiating wind puff are encoded by a population of several hundred broadly-tuned wind-sensitive hairs located on the bottom of the insect's cerci (two antennae-like structures found at the rear of the animal). The sensory neurons which innervate these hairs project to a small population of four pairs of ventral giant interneurons (the vGIs). These giant interneurons excite a larger population of approximately 100 interneurons located in the thoracic ganglia associated with each pair of legs. These type A thoracic interneurons (the $TI_As$) integrate information from a variety of other sources as well, including leg proprioceptors. Finally, the $TI_As$ project to local interneurons and motor neurons responsible for the control of each leg.

Perhaps what is most interesting about this system is that, despite the complexity of the response it controls, and despite the fact that its operation appears to be distributed across several populations of interneurons, the individual members of these populations are uniquely identifiable. For this reason, we believe that the cockroach escape response is an excellent model system for exploring the neuronal basis of distributed sensorimotor control at the level of identified nerve cells. As an integral part of that effort, we are constructing a computer model of the cockroach escape response.

## 3   NEURAL NETWORK MODEL

While a great deal is known about the overall response properties of many of the individual neurons in the escape circuit, as well as their architecture of connectivity, little detailed biophysical data is currently available. For this reason, our initial models have employed simplified neural network models and learning techniques. This approach has proven to be effective for analyzing a variety of neuronal circuits (e.g. Lockery *et al.*, 1989; Anastasio and Robinson, 1989). Specifically, using backpropagation, we train model neurons to reproduce the observed properties of identified nerve cells in the escape circuit.

In order to ensure that the resulting models are biologically relevant, we constrain

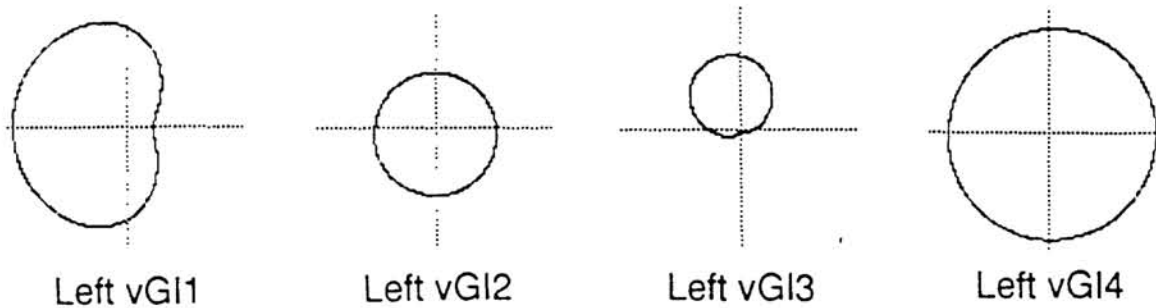

Left vGI1          Left vGI2          Left vGI3          Left vGI4

Figure 1: Windfields of Left Model Ventral Giant Interneurons

backpropagation to produce solutions which are consistent with the known structural characteristics of the circuit. The most important constraints we have utilized to date are the existence or nonexistence of specific connections between identified cells and the signs of existing connections. Other constraints that we are exploring include the firing curves and physiological operating ranges of identified neurons in the circuit. It is important to emphasize that we employed backpropagation solely as a means for finding the appropriate connection weights given the known structure of the circuit, and no claim is being made about its biological validity.

As an example of this approach, we have reconstructed the observed windfields of the eight ventral giant interneurons which serve as the first stage of interneuronal processing in the escape circuit. These windfields, which represent the intensity of a cell's response to wind puffs from different directions, have been well characterized in the insect (Westin, Langberg, and Camhi, 1977). The windfields of individual cercal sensory neurons have also been mapped (Westin, 1979; Daley and Camhi, 1988). The response of each hair is broadly tuned about a single preferred direction, which we have modeled as a cardioid. The cercal hairs are arranged in nine major columns on each cercus. All of the hairs in a single column share similar responses. Together, the responses of the hairs in all eighteen columns provide overlapping coverage of most directions around the insect's body. The connectivity between each major cercal hair column and each ventral giant interneuron is known, as are the signs of these connections (Daley and Camhi, 1988). Using these data, each model vGI was trained to reproduce the corresponding windfield by constrained backpropagation.[1] The resulting responses of the left four model vGIs are shown in Figure 1. These model windfields closely approximate those observed in the cockroach. Further details concerning vGI windfield reconstruction will be given in a forthcoming paper.

## 4   ESCAPE TURN RECONSTRUCTIONS

Ultimately, we are interested in simulating the entire escape response. This requires some way to connect our neural models to behavior, an approach that we have termed *computational neuroethology* (Beer, 1990). Toward that end, we have

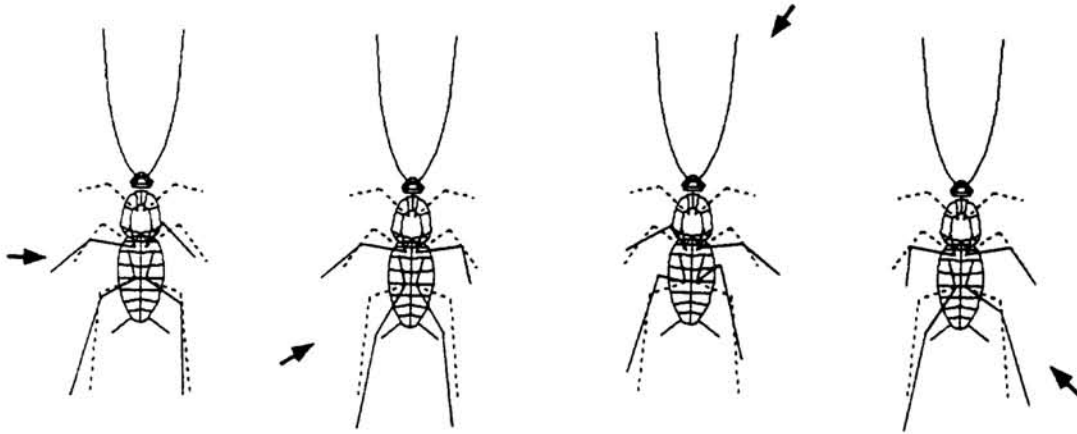

Figure 2: Model Escape Turns for Wind from Different Directions

also constructed a three dimensional kinematic model of the insect's body which accurately represents the essential degrees of freedom of the legs during escape turns. For our purposes here, the essential joints are the coxal-femur (CF) and femur-tibia (FT) joints of each leg. The leg segment lengths and orientations, as well as the joint angles and axes of rotation, were derived from actual measurements (Nye and Ritzmann, unpublished data). The active leg movements during escape turns of a tethered insect, in which the animal is suspended by a rod above a greased plate, have been shown to be identical to those of a free ranging animal (Camhi and Levy, 1988). Because an insect thus tethered is neither supporting its own weight nor generating appreciable forces with its legs, a kinematic body model can be defended as an adequate first approximation.

The leg movements of the simulated body were controlled by a neural network model of the entire escape circuit. Where sufficient data was available, the structure of this network was constrained appropriately. The first layer of this circuit was described in the previous section and is prevented from further training here. There are six groups of six representative $TI_A$s, one group for each leg. Within a group, representative members of each identified class of $TI_A$ are modeled. Where known, the connectivity from the vGIs to each class of $TI_A$s was enforced and all connections from vGIs to $TI_A$s were constrained to be excitatory (Ritzmann and Pollack, 1988). Model $TI_A$s also receive inputs from leg proprioceptors which encode the angle of each joint (Murrain and Ritzmann, 1988). The $TI_A$ layer for each side of the body was fully connected to 12 local interneurons, which were in turn fully connected to motor neurons which encode the change in angle of each joint in the body model.

High speed video films of the leg movements underlying actual escape turns in the tethered preparation for a variety of different wind angles and initial joint angles have been made (Nye and Ritzmann, 1990). The angles of each joint before wind onset and immediately after completion of the initial turn were used as training data for the model escape circuit. Only movements of the middle and hind legs were considered because individual joint angles of the front legs were far more variable. After training with constrained backpropagation, the model successfully reproduced the essential features of this data (Figure 2). Wind from the rear always caused

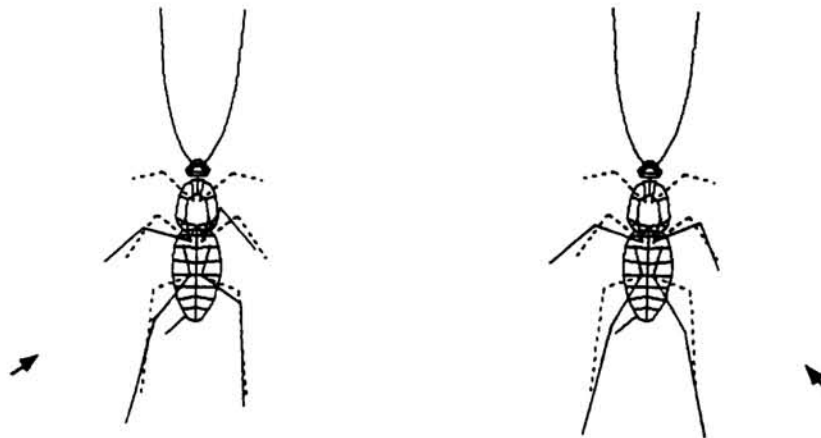

Figure 3: Model Escape Turns Following Left Cercal Ablation

the rear legs of the model to thrust back, which would propel the body forward in a freely moving insect, while wind from the front caused the rear legs to move forward, pulling the body back. The middle legs always turned the body away from the direction of the wind.

## 5  MODEL MANIPULATIONS

The results described above demonstrate that several neuronal and behavioral properties of this system can be reproduced using only simplified but biologically constrained neural network models. However, to serve as a useful tool for understanding the neuronal basis of the cockroach escape response, it is not enough for the model to simply reproduce what is already known about the normal operation of the system. In order to test and refine the model, we must also examine its responses to various lesions and compare them to the responses of the insect to analogous lesions. Here we report the results of two experiments of this sort.

Immediately following removal of the left cercus, cockroaches make a much higher proportion of incorrect turns (i.e. turns toward rather than away from the wind source) in response to wind from the left, while turns in response to wind from the right are largely unaffected. (Vardi and Camhi, 1982a). These results suggest that, despite the redundant representation of wind direction by each cercus, the insect integrates information from *both* cerci in order to compute the appropriate direction of movement. As shown in Figure 3, the response of the model to a left cercal ablation is consistent with these results. In response to wind from the unlesioned side, the model generates leg movements which would turn the body away from the wind. However, in response to wind from the lesioned side, the model generates leg movements which would turn the body toward the wind.

It is interesting to note that, following an approximately thirty day recovery period, the directionality of a cercally ablated cockroach's escape response is largely restored (Vardi and Camhi, 1982a). While the mechanisms underlying this adaptation are not yet fully understood, they appear to involve a reorganization of the vGI connections from the intact cercus (Vardi and Camhi, 1982b). After a cercal

ablation, the windfields of the vGIs on the ablated side are significantly reduced. Following the thirty day recovery period, however, these windfields are largely restored. We have also examined these effects in the model. After cercal ablation, the model vGI windfields show some similarities to those of similarly lesioned insects. In addition, using vGI retraining to simulate the adaptation process, we have found that the model can effect a similar recovery of vGI windfields by adjusting the connections from the intact cercus. However, due to space limitations, these results will be described in detail elsewhere.

A second experimental manipulation that has been performed on this system is the selective lesion of individual ventral giant interneurons (Comer, 1985). The only result that we will describe here is the lesion of vGI1. In the animal, this results in a behavioral deficit similar to that observed with cercal ablation. Correct turns result for wind from the unlesioned side, but a much higher proportion of incorrect turns are observed in response to wind from the lesioned side. The response of the model to this lesion is also similar to its response to a cercal ablation (Figure 3) and is thus consistent with these experimental results.

## 6    CONCLUSIONS

With the appropriate caveats, invertebrate systems offer the possibility of addressing important neurobiological questions at a much finer level than is generally possible in mammalian systems. In particular, the cockroach escape response is a complex sensorimotor control system whose operation is distributed across several populations of interneurons, but is nevertheless amenable to a detailed cellular analysis. Due to the overall complexity of such circuits and the wealth of data which can be extracted from them, modeling must play a crucial role in this endeavor. However, in order to be useful, models must make special efforts to remain consistent with known biological data and constantly be subjected to experimental test. Experimental work in turn must be responsive to model demands and predictions. This paper has described our initial results with this cooperative approach to the cockroach escape response. Our future work will focus on extending the current model in a similar manner.

### Acknowledgements

This work was supported by ONR grant N00014-90-J-1545 to RDB, a CAISR graduate fellowship from the Cleveland Advanced Manufacturing Program to GJK, NIH grant NS 17411 to RER, and NSF grant BNS-8810757 to HJC.

## Footnotes

[1]Strictly speaking, we are only using the delta rule here. The full power of backpropagation is not needed for this task since we are training only a single layer of weights.

### References

Anastasio, T.J. and Robinson, D.A. (1989). Distributed parallel processing in the vestibulo-oculomotor system. *Neural Computation* 1:230-241.

Beer, R.D. (1990). *Intelligence as Adaptive Behavior: An Experiment in Computational Neuroethology*. Academic Press.

Camhi, J.M. and Levy, A. (1988). Organization of a complex motor act: Fixed and variable components of the cockroach escape behavior. *J. Comp. Physiology*

163:317-328.

Comer, C.M. (1985). Analyzing cockroach escape behavior with lesions of individual giant interneurons. *Brain Research* **335**:342-346.

Daley, D.L. and Camhi, J.M. (1988). Connectivity pattern of the cercal-to-giant interneuron system of the American cockroach. *J. Neurophysiology* **60**:1350-1368.

Georgopoulos, A.P., Kettner, R.E. and Schwartz, A.B. (1988). Primate motor cortex and free arm movements to visual targets in three-dimensional space. II. Coding of the direction of movement by a neuronal population. *J. Neuroscience* **8**:2928-2937.

Lee, C., Rohrer, W.H. and Sparks, D.L. (1988). Population coding of saccadic eye movements by neurons in the superior colliculus. *Nature* **332**:357-360.

Lockery, S.R., Wittenberg, G., Kristan, W.B. Jr. and Cottrell, G.W. (1989). Function of identified interneurons in the leech elucidated using neural networks trained by back-propagation. *Nature* **340**:468-471.

Murrain, M. and Ritzmann, R.E. (1988). Analysis of proprioceptive inputs to DPG interneurons in the cockroach. *J. Neurobiology* **19**:552-570.

Nye, S.W. and Ritzmann, R.E. (1990). Videotape motion analysis of leg joint angles during escape turns of the cockroach. *Society for Neurosciences Abstracts* **16**:759.

Ritzmann, R.E. (1984). The cockroach escape response. In R.C. Eaton (Ed.) *Neural Mechanisms of Startle Behavior* (pp. 93-131). New York: Plenum.

Ritzmann, R.E. and Pollack, A.J. (1988). Wind activated thoracic interneurons of the cockroach: II. Patterns of connection from ventral giant interneurons. *J. Neurobiology* **19**:589-611.

Ritzmann, R.E. and Pollack, A.J. (1990). Parallel motor pathways from thoracic interneurons of the ventral giant interneuron system of the cockroach, *Periplaneta americana*. *J. Neurobiology* **21**:1219-1235.

Ritzmann, R.E., Pollack, A.J., Hudson, S. and Hyvonen, A. (in preparation). Thoracic interneurons in the escape system of the cockroach, *Periplaneta americana*, are multi-modal interneurons.

Vardi, N. and Camhi, J.M. (1982). Functional recovery from lesions in the escape system of the cockroach. I. Behavioral recovery. *J. Comp. Physiology* **146**:291-298.

Vardi, N. and Camhi, J.M. (1982). Functional recovery from lesions in the escape system of the cockroach. II. Physiological recovery of the giant interneurons. *J. Comp. Physiology* **146**:299-309.

Westin, J. (1979). Responses to wind recorded from the cercal nerve of the cockroach *Periplaneta americana*. I. Response properties of single sensory neurons. *J. Comp. Physiology* **133**:97-102.

Westin, J., Langberg, J.J. and Camhi, J.M. (1977). Response properties of giant interneurons of the cockroach *Periplaneta americana* to wind puffs of different directions and velocities. *J. Comp. Physiology* **121**:307-324.